# A Precise Characterization of the Class of Languages Recognized by Neural Nets under Gaussian and other Common Noise Distributions

**Wolfgang Maass**[*]
Inst. for Theoretical Computer Science,
Technische Universität Graz
Klosterwiesgasse 32/2,
A-8010 Graz, Austria
email: maass@igi.tu-graz.ac.at

**Eduardo D. Sontag**
Dep. of Mathematics
Rutgers University
New Brunswick, NJ 08903, USA
email: sontag@hilbert.rutgers.edu

## Abstract

We consider recurrent analog neural nets where each gate is subject to Gaussian noise, or any other common noise distribution whose probability density function is nonzero on a large set. We show that many regular languages cannot be recognized by networks of this type, for example the language $\{w \in \{0,1\}^* | \; w \text{ begins with } 0\}$, and we give a precise characterization of those languages which can be recognized. This result implies severe constraints on possibilities for constructing recurrent analog neural nets that are robust against realistic types of analog noise. On the other hand we present a method for constructing *feedforward* analog neural nets that are robust with regard to analog noise of this type.

## 1   Introduction

A fairly large literature (see [Omlin, Giles, 1996] and the references therein) is devoted to the construction of analog neural nets that recognize regular languages. Any physical realization of the analog computational units of an analog neural net in technological or biological systems is bound to encounter some form of "imprecision" or analog noise at its analog computational units. We show in this article that this effect has serious consequences for the computational power of recurrent analog neural nets. We show that any analog neural net whose computational units are subject to Gaussian or other common noise distributions cannot recognize arbitrary regular languages. For example, such analog neural net cannot recognize the regular language $\{w \in \{0,1\}^* | \; w \text{ begins with } 0\}$.

---

[*]Partially supported by the Fonds zur Förderung der wissenschaftlichen Forschung (FWF), Austria, project P12153.

A precise characterization of those regular languages which can be recognized by such analog neural nets is given in Theorem 1.1. In section 3 we introduce a simple technique for making feedforward neural nets robust with regard to the same types of analog noise. This method is employed to prove the positive part of Theorem 1.1. The main difficulty in proving Theorem 1.1 is its negative part, for which adequate theoretical tools are introduced in section 2.

Before we can give the exact statement of Theorem 1.1 and discuss related preceding work we have to give a precise definition of computations in *noisy* neural networks. From the conceptual point of view this definition is basically the same as for computations in noisy boolean circuits (see [Pippenger, 1985] and [Pippenger, 1990]). However it is technically more involved since we have to deal here with an infinite state space.

We will first illustrate this definition for a concrete case, a recurrent sigmoidal neural net with Gaussian noise, and then indicate the full generality of our result, which makes it applicable to a very large class of other types of analog computational systems with analog noise. Consider a recurrent sigmoidal neural net $\mathcal{N}$ consisting of $n$ units, that receives at each time step $t$ an input $u_t$ from some finite alphabet $U$ (for example $U = \{0,1\}$). The internal state of $\mathcal{N}$ at the end of step $t$ is described by a vector $x_t \in [-1,1]^n$, which consists of the outputs of the $n$ sigmoidal units at the end of step $t$. A computation step of the network $\mathcal{N}$ is described by

$$x_{t+1} = \sigma(Wx_t + h + u_tc + V_t)$$

where $W \in \mathbb{R}^{n \times n}$ and $c, h \in \mathbb{R}^n$ represent weight matrix and vectors, $\sigma$ is a sigmoidal activation function (e.g., $\sigma(y) = 1/(1 + e^{-y})$) applied to each vector component, and $V_1, V_2, \ldots$ is a sequence of $n$-vectors drawn independently from some Gaussian distribution. In analogy to the case of noisy boolean circuits [Pippenger, 1990] one says that this network $\mathcal{N}$ *recognizes a language* $L \subseteq U^*$ *with reliability* $\varepsilon$ (where $\varepsilon \in (0, \frac{1}{2}]$ is some given constant) if immediately after reading an arbitrary word $w \in U^*$ the network $\mathcal{N}$ is with probability $\geq \frac{1}{2} + \varepsilon$ in an accepting state in case that $w \in L$, and with probability $\leq \frac{1}{2} - \varepsilon$ in an accepting state in case that $w \notin L^1$.

We will show in this article that even if the parameters of the Gaussian noise distribution for each sigmoidal unit can be determined by the designer of the neural net, it is impossible to find a size $n$, weight matrix $W$, vectors $h, c$ and a reliability $\varepsilon \in (0, \frac{1}{2}]$ so that the resulting recurrent sigmoidal neural net with Gaussian noise accepts the simple regular language $\{w \in \{0,1\}^* \mid w \text{ begins with } 0\}$ with reliability $\varepsilon$. This result exhibits a fundamental limitation for making a recurrent analog neural net noise robust, even in a case where the noise distribution is known and of a rather benign type. This quite startling negative result should be contrasted with the large number of known techniques for making a feedforward boolean circuit robust against noise, see [Pippenger, 1990].

Our negative result turns out to be of a very general nature, that holds for virtually all related definitions of noisy analog neural nets and also for completely different models for analog computation in the presence of Gaussian or similar noise. Instead of the state set $[-1,1]^n$ one can take any compact set $\Omega \subseteq \mathbb{R}^n$, and instead of the map $(x,u) \mapsto Wx + h + uc$ one can consider an arbitrary map $f : \Omega \times U \to \widehat{\Omega}$ for a compact set $\widehat{\Omega} \subseteq \mathbb{R}^n$ where $f(\cdot, u)$ is Borel measurable for each fixed $u \in U$. Instead of a sigmoidal activation function $\sigma$ and a Gaussian distributed noise vector $V$ it suffices to assume that $\sigma : \mathbb{R}^n \to \Omega$ is some arbitrary Borel measurable function and $V$ is some $\mathbb{R}^n$-valued random variable with a density $\phi(\cdot)$ that has a wide support[2]. In order to define a computation in such system we consider for

each $u \in U$ the stochastic kernel $K_u$ defined by $K_u(x, A) := \text{Prob}\left[\sigma(f(x, u) + V) \in A\right]$ for $x \in \Omega$ and $A \subseteq \Omega$. For each (signed, Borel) measure $\mu$ on $\Omega$, and each $u \in U$, we let $\mathbb{K}_u\mu$ be the (signed, Borel) measure defined on $\Omega$ by $(\mathbb{K}_u\mu)(A) := \int K_u(x, A)d\mu(x)$. Note that $\mathbb{K}_u\mu$ is a probability measure whenever $\mu$ is. For any sequence of inputs $w = u_1, \ldots, u_r$, we consider the composition of the evolution operators $\mathbb{K}_{u_i}$:

$$\mathbb{K}_w = \mathbb{K}_{u_r} \circ \mathbb{K}_{u_{r-1}} \circ \ldots \circ \mathbb{K}_{u_1}. \tag{1}$$

If the probability distribution of states at any given instant is given by the measure $\mu$, then the distribution of states after a single computation step on input $u \in U$ is given by $\mathbb{K}_u\mu$, and after $r$ computation steps on inputs $w = u_1, \ldots, u_r$, the new distribution is $\mathbb{K}_w\mu$, where we are using the notation (1). In particular, if the system starts at a particular initial state $\xi$, then the distribution of states after $r$ computation steps on $w$ is $\mathbb{K}_w\delta_\xi$, where $\delta_\xi$ is the probability measure concentrated on $\{\xi\}$. That is to say, for each measurable subset $F \subseteq \Omega$

$$\text{Prob}\left[x_{r+1} \in F \mid x_1 = \xi, \text{ input } = w\right] = (\mathbb{K}_w\delta_\xi)(F).$$

We fix an initial state $\xi \in \Omega$, a set of "accepting" or "final" states $F$, and a "reliability" level $\varepsilon > 0$, and say that the resulting noisy analog computational system $M$ *recognizes the language* $L \subseteq U^*$ if for all $w \in U^*$:

$$w \in L \iff (\mathbb{K}_w\delta_\xi)(F) \geq \frac{1}{2} + \varepsilon$$

$$w \notin L \iff (\mathbb{K}_w\delta_\xi)(F) \leq \frac{1}{2} - \varepsilon.$$

In general a neural network that simulates a DFA will carry out not just one, but a fixed number $k$ of computation steps (=state transitions) of the form $x' = \sigma(Wx + h + uc + V)$ for each input symbol $u \in U$ that it reads (see the constructions described in [Omlin, Giles, 1996], and in section 3 of this article). This can easily be reflected in our model by formally replacing any input sequence $w = u_1, u_2, \ldots, u_r$ from $U^*$ by a padded sequence $\tilde{w} = u_1, b^{k-1}, u_2, b^{k-1}, \ldots, u_r, b^{k-1}$ from $(U \cup \{b\})^*$, where $b$ is a blank symbol not in $U$, and $b^{k-1}$ denotes a sequence of $k - 1$ copies of $b$ (for some arbitrarily fixed $k \geq 1$). This completes our definition of language recognition by a noisy analog computational system $M$ with discrete time. This definition essentially agrees with that given in [Maass, Orponen, 1997].

We employ the following common notations from formal language theory: We write $w_1w_2$ for the concatenation of two strings $w_1$ and $w_2$, $U^r$ for the set of all concatenations of $r$ strings from $U$, $U^*$ for the set of all concatenations of any finite number of strings from $U$, and $UV$ for the set of all strings $w_1w_2$ with $w_1 \in U$ and $w_2 \in V$. The main result of this article is the following:

**Theorem 1.1** *Assume that $U$ is some arbitrary finite alphabet. A language $L \subseteq U^*$ can be recognized by a noisy analog computational system of the previously specified type if and only if $L = E_1 \bigcup U^*E_2$ for two finite subsets $E_1$ and $E_2$ of $U^*$.*

A corresponding version of Theorem 1.1 for *discrete* computational systems was previously shown in [Rabin, 1963]. More precisely, Rabin had shown that probabilistic automata with strictly positive matrices can recognize exactly the same class of languages $L$ that occur in our Theorem 1.1. Rabin referred to these languages as *definite languages*. Language recognition by *analog* computational systems with *analog* noise has previously been investigated in [Casey, 1996] for the special case of bounded noise and perfect reliability

---

that the following two properties hold: $\phi(v) \geq c_0$ for all $v \in Q := \sigma^{-1}(\Omega_0) - \widehat{\Omega}$ (that is, $Q$ is the set consisting of all possible differences $z - y$, with $\sigma(z) \in \Omega_0$ and $y \in \widehat{\Omega}$) *and* $\sigma^{-1}(\Omega_0)$ has finite and nonzero Lebesgue measure $m_0 = \lambda\left(\sigma^{-1}(\Omega_0)\right)$.

(i.e. $\int_{\|v\|\leq\eta}\phi(v)dv=1$ for some small $\eta>0$ and $\varepsilon=1/2$ in our terminology), and in [Maass, Orponen, 1997] for the general case. It was shown in [Maass, Orponen, 1997] that any such system can only recognize regular languages. Furthermore it was shown there that if $\int_{\|v\|\leq\eta}\phi(v)dv=1$ for some small $\eta>0$ then all regular languages *can* be recognized by such systems. In the present paper we focus on the complementary case where the condition "$\int_{\|v\|\leq\eta}\phi(v)dv=1$ for some small $\eta>0$" is not satisfied, i.e. analog noise may move states over larger distances in the state space. We show that even if the probability of such event is arbitrarily small, the neural net will no longer be able to recognize arbitrary regular languages.

## 2   A Constraint on Language Recognition

We prove in this section the following result for arbitrary noisy computational systems $M$ as specified at the end of section 1:

**Theorem 2.1** *Assume that $U$ is some arbitrary alphabet. If a language $L \subseteq U^*$ is recognized by $M$, then there are subsets $E_1$ and $E_2$ of $U^{\leq r}$, for some integer $r$, such that $L = E_1 \bigcup U^* E_2$. In other words: whether a string $w \in U^*$ belongs to the language $L$ can be decided by just inspecting the first $r$ and the last $r$ symbols of $w$.*

### 2.1   A General Fact about Stochastic Kernels

Let $(S, \mathcal{S})$ be a measure space, and let $K$ be a stochastic kernel[3]. As in the special case of the $K_u$'s above, for each (signed) measure $\mu$ on $(S, \mathcal{S})$, we let $\mathbb{K}\mu$ be the (signed) measure defined on $\mathcal{S}$ by $(\mathbb{K}\mu)(A) := \int K(x, A)d\mu(x)$. Observe that $\mathbb{K}\mu$ is a probability measure whenever $\mu$ is. Let $c > 0$ be arbitrary. We say that $K$ satisfies *Doeblin's condition (with constant c)* if there is some probability measure $\rho$ on $(S, \mathcal{S})$ so that

$$K(x, A) \geq c\rho(A) \quad \text{for all } x \in S, A \in \mathcal{S}. \tag{2}$$

(Necessarily $c \leq 1$, as is seen by considering the special case $A = S$.) This condition is due to [Doeblin, 1937].

We denote by $\|\mu\|$ the *total variation* of the (signed) measure $\mu$. Recall that $\|\mu\|$ is defined as follows. One may decompose $S$ into a disjoint union of two sets $A$ and $B$, in such a manner that $\mu$ is nonnegative on $A$ and nonpositive on $B$. Letting the restrictions of $\mu$ to $A$ and $B$ be "$\mu_+$" and "$-\mu_-$" respectively (and zero on $B$ and $A$ respectively), we may decompose $\mu$ as a difference of nonnegative measures with disjoint supports, $\mu = \mu_+ - \mu_-$. Then, $\|\mu\| = \mu_+(A) + \mu_-(B)$. The following Lemma is a "folk" fact ([Papinicolaou, 1978]).

**Lemma 2.2** *Assume that $K$ satisfies Doeblin's condition with constant c. Let $\mu$ be any (signed) measure such that $\mu(S) = 0$. Then $\|\mathbb{K}\mu\| \leq (1-c)\|\mu\|$.* ∎

### 2.2   Proof of Theorem 2.1

**Lemma 2.3** *There is a constant $c > 0$ such that $K_u$ satisfies Doeblin's condition with constant c, for every $u \in U$.*

*Proof.* Let $\Omega_0$, $c_0$, and $0 < m_0 < 1$ be as in the second footnote, and introduce the following (Borel) probability measure on $\Omega_0$:

$$\lambda_0(A) := \frac{1}{m_0}\lambda\left(\sigma^{-1}(A)\right).$$

Pick any measurable $A \subseteq \Omega_0$ and any $y \in \widehat{\Omega}$. Then,

$$
\begin{aligned}
Z(y, A) &= \text{Prob}\left[\sigma(y + V) \in A\right] = \text{Prob}\left[y + V \in \sigma^{-1}(A)\right] \\
&= \int_{A_y} \phi(v)\, dv \geq c_0 \lambda(A_y) = c_0 \lambda\left(\sigma^{-1}(A)\right) = c_0 m_0 \lambda_0(A),
\end{aligned}
$$

where $A_y := \sigma^{-1}(A) - \{y\} \subseteq Q$. We conclude that $Z(y, A) \geq c\lambda_0(A)$ for all $y, A$, where $c = c_0 m_0$. Finally, we extend the measure $\lambda_0$ to all of $\Omega$ by assigning zero measure to the complement of $\Omega_0$, that is, $\rho(A) := \lambda_0(A \cap \Omega_0)$ for all measurable subsets $A$ of $\Omega$. Pick $u \in U$; we will show that $K_u$ satisfies Doeblin's condition with the above constant $c$ (and using $\rho$ as the "comparison" measure in the definition). Consider any $x \in \Omega$ and measurable $A \subseteq \Omega$. Then,

$$
K_u(x, A) = Z(f(x, u), A) \geq Z(f(x, u), A \cap \Omega_0) \geq c\lambda_0(A \cap \Omega_0) = c\rho(A),
$$

as required. ∎

For every two probability measures $\mu_1, \mu_2$ on $\Omega$, applying Lemma 2.2 to $\mu := \mu_1 - \mu_2$, we know that $\|K_u \mu_1 - K_u \mu_2\| \leq (1 - c)\|\mu_1 - \mu_2\|$ for each $u \in U$. Recursively, then, we conclude:

$$
\|K_w \mu_1 - K_w \mu_2\| \leq (1 - c)^r \|\mu_1 - \mu_2\| \leq 2(1 - c)^r \tag{3}
$$

for all words $w$ of length $\geq r$.

Now pick any integer $r$ such that $(1 - c)^r < 2\varepsilon$. From Equation (3), we have that

$$
\|K_w \mu_1 - K_w \mu_2\| < 4\varepsilon
$$

for all $w$ of length $\geq r$ and any two probability measures $\mu_1, \mu_2$. In particular, this means that, for each measurable set $A$,

$$
|(K_w \mu_1)(A) - (K_w \mu_2)(A)| < 2\varepsilon \tag{4}
$$

for all such $w$. (Because, for any two probability measures $\nu_1$ and $\nu_2$, and any measurable set $A$, $2|\nu_1(A) - \nu_2(A)| \leq \|\nu_1 - \nu_2\|$.)

**Lemma 2.4** *Pick any $v \in U^*$ and $w \in U^r$. Then*

$$
w \in L \iff vw \in L.
$$

*Proof.* Assume that $w \in L$, that is, $(K_w \delta_\xi)(F) \geq \frac{1}{2} + \varepsilon$. Applying inequality (4) to the measures $\mu_1 := \delta_\xi$ and $\mu_2 := K_v \delta_\xi$ and $A = F$, we have that $|(K_w \delta_\xi)(F) - (K_{vw} \delta_\xi)(F)| < 2\varepsilon$, and this implies that $(K_{vw} \delta_\xi)(F) > \frac{1}{2} - \varepsilon$, i.e., $vw \in L$. (Since $\frac{1}{2} - \varepsilon < (K_{vw} \delta_\xi)(F) < \frac{1}{2} + \varepsilon$ is ruled out.) If $w \notin L$, the argument is similar. ∎

We have proved that

$$
L \cap (U^* U^r) = U^*(L \cap U^r).
$$

So,

$$
L = \left(L \cap U^{\leq r}\right) \cup \left(L \cap U^* U^r\right) = E_1 \cup U^* E_2
$$

where $E_1 := L \cap U^{\leq r}$ and $E_2 := L \cap U^r$ are both included in $U^{\leq r}$. This completes the proof of Theorem 2.1. ∎

## 3   Construction of Noise Robust Analog Neural Nets

In this section we exhibit a method for making feedforward analog neural nets robust with regard to arbitrary analog noise of the type considered in the preceding sections. This method will be used to prove in Corollary 3.2 the missing positive part of the claim of the main result (Theorem 1.1) of this article.

**Theorem 3.1** *Let $C$ be any (noiseless) feedforward threshold circuit, and let $\sigma : \mathbb{R} \to [-1, 1]$ be some arbitrary function with $\sigma(u) \to 1$ for $u \to \infty$ and $\sigma(u) \to -1$ for $u \to -\infty$. Furthermore assume that $\delta, \rho \in (0, 1)$ are some arbitrary given parameters. Then one can transform for any given analog noise of the type considered in section 1 the noiseless threshold circuit $C$ into an analog neural net $\mathcal{N}_C$ with the same number of gates, whose gates employ the given function $\sigma$ as activation function, so that for any circuit input $\underline{x} \in \{-1, 1\}^m$ the output of the noisy analog neural net $\mathcal{N}_C$ differs with probability $\geq 1 - \delta$ by at most $\rho$ from the output of $C$.*

Idea of the *proof:* Let $k$ be the maximal fan-in of a gate in $C$, and let $w$ be the maximal absolute value of a weight in $C$. We choose $R > 0$ so large that the density function $\phi(\cdot)$ of the noise vector $V$ satisfies for each gate with $n$ inputs in $C$

$$\int_{|v_i| \geq R} \phi(v)\, dv \;\leq\; \frac{\delta}{2n} \;\text{ for } i = 1, \ldots, n.$$

Furthermore we choose $u_0 > 0$ so large that $\sigma(u) \geq 1 - \rho/(wk)$ for $u \geq u_0$ and $\sigma(u) \leq -1 + \rho/(wk)$ for $u \leq -u_0$. Finally we choose a factor $\gamma > 0$ so large that $\gamma(1-\rho) - R \geq u_0$. Let $\mathcal{N}_C$ be the analog neural net that results from $C$ through multiplication of all weights and thresholds with $\gamma$ and through replacement of the Heaviside activation functions of the gates in $C$ by the given activation function $\sigma$.                                                                       ∎

The following Corollary provides the proof of the positive part of our main result Theorem 1.1. It holds for any $\sigma$ considered in Theorem 3.1.

**Corollary 3.2** *Assume that $U$ is some arbitrary finite alphabet, and language $L \subseteq U^*$ is of the form $L = E_1 \bigcup U^* E_2$ for two arbitrary finite subsets $E_1$ and $E_2$ of $U^*$. Then the language $L$ can be recognized by a noisy analog neural net $\mathcal{N}$ with any desired reliability $\varepsilon \in (0, \frac{1}{2})$, in spite of arbitrary analog noise of the type considered in section 1.*

*Proof.* We first construct a feedforward threshold circuit $C$ for recognizing $L$, that receives each input symbol from $U$ in the form of a bitstring $u \in \{0, 1\}^l$ (for some fixed $l \geq \log_2 |U|$), that is encoded as the binary states of $l$ input units of the boolean circuit $C$. Via a tapped delay line of fixed length $d$ (which can easily be implemented in a feedforward threshold circuit by $d$ layers, each consisting of $l$ gates that compute the identity function on a single binary input from the preceding layer) one can achieve that the feedforward circuit $C$ computes *any* given boolean function of the last $d$ sequences from $\{0, 1\}^l$ that were presented to the circuit. On the other hand for any language of the form $L = E_1 \cup U^* E_2$ with $E_1, E_2$ finite there exists some $d \in \mathbb{N}$ such that for each $w \in U^*$ one can decide whether $w \in L$ by just inspecting the last $d$ characters of $w$. Therefore a feedforward threshold circuit $C$ with a tapped delay line of the type described above can decide whether $w \in L$.

We apply Theorem 3.1 to this circuit $C$ for $\delta = \rho = \min(\frac{1}{2} - \varepsilon, \frac{1}{4})$. We define the set F of accepting states for the resulting analog neural net $\mathcal{N}_C$ as the set of those states where the computation is completed and the output gate of $\mathcal{N}_C$ assumes a value $\geq 3/4$. Then according to Theorem 3.1 the analog neural net $\mathcal{N}_C$ recognizes $L$ with reliability $\varepsilon$. To be formally precise, one has to apply Theorem 3.1 to a threshold circuit $C$ that receives its

input not in a single batch, but through a sequence of $d$ batches. The proof of Theorem 3.1 readily extends to this case. ∎

## 4 Conclusions

We have exhibited a fundamental limitation of analog neural nets with Gaussian or other common noise distributions whose probability density function is nonzero on a large set: They cannot accept the very simple regular language $\{w \in \{0,1\}^* \mid w \text{ begins with } 0\}$. This holds even if the designer of the neural net is allowed to choose the parameters of the Gaussian noise distribution and the architecture and parameters of the neural net. The proof of this result introduces new mathematical arguments into the investigation of neural computation, which can also be applied to other stochastic analog computational systems.

We also have presented a method for making *feedforward* analog neural nets robust against the same type of noise. This implies that certain regular languages, such as for example $\{w \in \{0,1\}^* \mid w \text{ ends with } 0\}$ *can* be recognized by a recurrent analog neural net with Gaussian noise. In combination with our negative result this yields a *precise characterization* of all regular languages that can be recognized by recurrent analog neural nets with Gaussian noise, or with any other noise distribution that has a large support.

## Footnotes

[1]According to this definition a network $\mathcal{N}$ that is after reading some $w \in U^*$ in an accepting state with probability strictly between $\frac{1}{2} - \varepsilon$ and $\frac{1}{2} + \varepsilon$ does not recognize any language $L \subseteq U^*$.

[2]More precisely: We assume that there exists a subset $\Omega_0$ of $\Omega$ and some constant $c_0 > 0$ such

[3]That is to say, $K(x, \cdot)$ is a probability distribution for each $x$, and $K(\cdot, A)$ is a measurable function for each Borel measurable set $A$.

## References

[Casey, 1996] Casey, M., "The dynamics of discrete-time computation, with application to recurrent neural networks and finite state machine extraction", *Neural Computation* 8, 1135–1178, 1996.

[Doeblin, 1937] Doeblin, W., "Sur le propriétés asymtotiques de mouvement régis par certain types de chaînes simples", *Bull. Math. Soc. Roumaine Sci. 39(1)*: 57–115; (2) 3–61, 1937.

[Maass, Orponen, 1997] Maass, W., and Orponen, P. "On the effect of analog noise on discrete-time analog computations", *Advances in Neural Information Processing Systems 9*, 1997, 218–224; journal version: *Neural Computation* 10(5), 1071–1095, 1998.

[Omlin, Giles, 1996] Omlin, C. W., Giles, C. L. "Constructing deterministic finite-state automata in recurrent neural networks", *J. Assoc. Comput. Mach. 43* (1996), 937–972.

[Papinicolaou, 1978] Papinicolaou, G., "Asymptotic Analysis of Stochastic Equations", in *Studies in Probability Theory, MAA Studies in Mathematics*, vol. 18, 111–179, edited by M. Rosenblatt, Math. Assoc. of America, 1978.

[Pippenger, 1985] Pippenger, N., "On networks of noisy gates", *IEEE Sympos. on Foundations of Computer Science*, vol. 26, IEEE Press, New York, 30–38, 1985.

[Pippenger, 1989] Pippenger, N., "Invariance of complexity measures for networks with unreliable gates", *J. of the ACM*, vol. 36, 531–539, 1989.

[Pippenger, 1990] Pippenger, N., "Developments in 'The Synthesis of Reliable Organisms from Unreliable Components' ", *Proc. of Symposia in Pure Mathematics*, vol. 50, 311–324, 1990.

[Rabin, 1963] Rabin, M., "Probabilistic automata", *Information and Control*, vol. 6, 230–245, 1963.